# How Linear are Auditory Cortical Responses?

**Maneesh Sahani**
Gatsby Unit, UCL
17 Queen Sq., London, WC1N 3AR, UK.
maneesh@gatsby.ucl.ac.uk

**Jennifer F. Linden**
Keck Center, UCSF
San Francisco, CA 94143–0732.
linden@phy.ucsf.edu

## Abstract

By comparison to some other sensory cortices, the functional properties of cells in the primary auditory cortex are not yet well understood. Recent attempts to obtain a generalized description of auditory cortical responses have often relied upon characterization of the spectrotemporal receptive field (STRF), which amounts to a model of the stimulus-response function (SRF) that is linear in the spectrogram of the stimulus. How well can such a model account for neural responses at the very first stages of auditory cortical processing? To answer this question, we develop a novel methodology for evaluating the fraction of stimulus-related response power in a population that can be captured by a given type of SRF model. We use this technique to show that, in the thalamo-recipient layers of primary auditory cortex, STRF models account for no more than 40% of the stimulus-related power in neural responses.

## 1 Introduction

A number of recent studies have suggested that spectrotemporal receptive field (STRF) models [1, 2], which are linear in the stimulus spectrogram, can describe the spiking responses of auditory cortical neurons quite well [3, 4]. At the same time, other authors have pointed out significant non-linearities in auditory cortical responses [5, 6], or have emphasized both linear and non-linear response components [7, 8]. Some of the differences in these results may well arise from differences in the stimulus ensembles used to evoke neuronal responses. However, even for a single type of stimulus, it is extremely difficult to put a number to the proportion of the response that is linear or non-linear, and so to judge the relative contributions of the two components to the stimulus-evoked activity.

The difficulty arises because repeated presentations of identical stimulus sequences evoke highly variable responses from neurons at intermediate stages of perceptual systems, even in anaesthetized animals. While this variability may reflect meaningful changes in the internal state of the animal or may be completely random, from the point of view of modelling the relationship between stimulus and neural response it must be treated as noise. As previous authors have noted [9, 10], this noise complicates the evaluation of the performance of a particular class of stimulus-response function (SRF) model (for example, the class of STRF models) in two ways. First, it makes it difficult to assess the quality of the predictions given by any single model. Perfect prediction of a noisy response is impossible, even in principle, and since the the true underlying relationship between stimulus and neural response is unknown, it is unclear what degree of partial prediction could possibly

be expected. Second, the noise introduces error into the estimation of the model parameters; consequently, even where direct unbiased evaluations of the predictions made by the estimated models are possible, these evaluations understate the performance of the model in the class that most closely matches the true SRF.

The difficulties can be illustrated in the context of the classical statistical measure of the fraction of variance explained by a model, the coefficient of determination or $r^2$ statistic. This is the ratio of the reduction in variance achieved by the regression model (the total variance of the outputs minus the variance of the residuals) to the total variance of the outputs. The total variance of the outputs includes contributions from the noise, and so an $r^2$ of 1 is an unrealistic target, and the actual maximum achievable value is unclear. Moreover, the reduction of variance on the training data, which appears in the numerator of the $r^2$, includes some "explanation" of noise due to overfitting. The extent to which this happens is difficult to estimate; if the reduction in variance is evaluated on test data, estimation errors in the model will lead to an underestimate of the performance of the best model in the class. Hypothesis tests based on $r^2$ compensate for these shortcomings in answering questions of model sufficiency. However, these tests do not provide a way to assess the extent of partial validity of a model class; indeed, it is well known that even the failure of a hypothesis test to reject a specific model class is not sufficient evidence to regard the model as fully adequate. One proposed method for obtaining a more quantitative measure of model performance is to compare the correlation (or, equivalently, squared distance) between the model prediction and a new response measurement to that between two successive responses to the same stimulus [9, 11]; as acknowledged in those proposals, however, this yardstick underestimates the response reliability even after considerable averaging, and so the comparison will tend to overestimate the validity of the SRF model.

Measures like $r^2$ that are based on the fractional variance (or, for time series, the power) explained by a model do have some advantages; for example, contributions from independent sources are additive. Here, we develop analytic techniques that overcome the systematic noise-related biases in the usual variance measures[1], and thus obtain, for a population of neurons, a quantitative estimate of the fraction of stimulus-related response captured by a given class of models. This statistical framework may be applicable to analysis of response functions for many types of neural data, ranging from intracellular recordings to imaging measurements. We apply it to extracellular recordings from rodent auditory cortex, quantifying the degree to which STRF models can account for neuronal responses to dynamic random chord stimuli. We find that on average less than half of the reliable stimulus-related power in these responses can be captured by spectrogram-linear STRF models.

## 2   Signal power

The analysis assumes that the data consist of spike trains or other neural measurements continuously recorded during presentation of a long, complex, rapidly varying stimulus. This stimulus is treated as a discrete-time process. In the auditory experiment considered here, the discretization was set by the duration of regularly clocked sound pulses of fixed length; in a visual experiment, the discretization might be the frame rate of a movie. The neural response can then be measured with the same level of precision, counting action potentials (or integrating measurements) to estimate a response rate for each time bin, to obtain a response vector $\mathbf{r} = (r_t)_{t=1...T}$. We propose to measure model performance in terms of the fraction of *response power* predicted successfully, where "power" is used in the sense of average squared deviation from the mean: $P(\mathbf{r}) = \left\langle (r_t - \langle r_t \rangle)^2 \right\rangle$ ($\langle \cdot \rangle$ denoting

averages over time). As argued above, only some part of the total response power is predictable, even in principle; fortunately, this *signal power* can be estimated by combining repeated responses to the same stimulus sequence. We present a method-of-moments [12] derivation of the relevant estimator below.

Suppose we have $N$ responses $\mathbf{r}^{(n)} = \boldsymbol{\mu} + \boldsymbol{\eta}^{(n)}$, where $\boldsymbol{\mu}$ is the common, stimulus-dependent component (signal) in the response and $\boldsymbol{\eta}^{(n)}$ is the (zero-mean) noise component of the response in the $n$th trial. The expected power in each response is given by $P(\mathbf{r}^{(n)}) \overset{\mathcal{E}}{=} P(\boldsymbol{\mu}) + \left\langle (\eta_t^{(n)})^2 \right\rangle$ (where the symbol $\overset{\mathcal{E}}{=}$ means "equal in expectation"). This simple relationship depends only on the noise component having been defined to have zero mean, and holds even if the variance or other property of the noise depends on the signal strength. We now construct two trial-averaged quantities, similar to the sum-of-squares terms used in the analysis of variance (ANOVA) [12]: the power of the average response, and the average power per response. Using $\overline{\phantom{x}}$ to indicate trial averages:

$$P(\overline{\mathbf{r}^{(n)}}) \overset{\mathcal{E}}{=} P(\boldsymbol{\mu}) + P(\overline{\boldsymbol{\eta}^{(n)}}) \qquad \text{and} \qquad \overline{P(\mathbf{r}^{(n)})} \overset{\mathcal{E}}{=} P(\boldsymbol{\mu}) + \overline{P(\boldsymbol{\eta}^{(n)})}.$$

Assuming the noise in each trial is independent (although the noise in different time bins within a trial need not be), we have: $P(\overline{\boldsymbol{\eta}^{(n)}}) \overset{\mathcal{E}}{=} \overline{P(\boldsymbol{\eta}^{(n)})}/N$. Thus solving for $P(\boldsymbol{\mu})$ suggests the following estimator for the signal power:

$$\hat{P}(\boldsymbol{\mu}) = \frac{1}{N-1} \left( N P(\overline{\mathbf{r}^{(n)}}) - \overline{P(\mathbf{r}^{(n)})} \right). \tag{1}$$

(A similar estimator for the *noise power* is obtained by subtracting this expression from $\overline{P(\mathbf{r}^{(n)})}$.) This estimator is unbiased, provided only that the noise distribution has defined first and second moments and is independent between trials, as can be verified by explicitly calculating its expected value. Unlike the sum-of-squares terms encountered in an ANOVA, it is not a $\chi^2$ variate even when the noise is normally distributed (indeed, it is not necessarily positive). However, since each of the power terms in (1) is the mean of at least $T$ numbers, the central limit theorem suggests that $\hat{P}$ will be approximately normally distributed for recordings that are considerably longer than the time-scale of noise correlation (in the experiment considered here, $T = 3000$). Its variance is given by:

$$\mathcal{V}\left[\hat{P}\right] = \frac{4}{N} \left( \frac{1}{T^2}\boldsymbol{\mu}'\Sigma\boldsymbol{\mu} - \frac{2}{T}\mu\boldsymbol{\sigma}'\boldsymbol{\mu} + \mu\sigma\mu \right) + \frac{2}{N(N-1)} \left( \frac{1}{T^2}\mathrm{Tr}\left[\Sigma\Sigma\right] - \frac{2}{T}\boldsymbol{\sigma}'\boldsymbol{\sigma} + \sigma^2 \right),$$
$$\tag{2}$$

where $\Sigma$ is the $(T \times T)$ covariance matrix of the noise, $\boldsymbol{\sigma}$ is a vector formed by averaging each column of $\Sigma$, $\sigma$ is the average of all the elements of $\Sigma$ and $\mu$ is the time-average of the mean $\boldsymbol{\mu}$. Thus, $\mathcal{V}ar\left[\hat{P}\right]$ depends only on the first and second moments of the response distribution; substitution of data-derived estimates of these moments into (2) yields a standard error bar for the estimator. In this way we have obtained an estimate $\hat{P}$ (with corresponding uncertainty) of the maximum possible signal power that any model could accurately predict, without having assumed any particular distribution or time-independence of the noise.

## 3   Extrapolating Model Performance

To compare the performance of an estimated SRF model to this maximal value, we must determine the amount of response power successfully predicted by the model. This is not necessarily the power of the predicted response, since the prediction may be inaccurate. Instead, the residual power in the difference between a measured response $\mathbf{r}$ and the predicted response $\boldsymbol{\rho}$ to the same stimulus, $P(\mathbf{r} - \boldsymbol{\rho})$, is taken as an estimate of the error power. (The measured response used for this evaluation, and the stimulus which elicited it, may or may not also have been used to identify the parameters of the SRF model being evaluated; see explanation of training and test predictive powers below.) The difference between the

power in the observed response $P(\mathbf{r})$ and the error power gives the *predictive power* of the model; it is this value that can be compared to the estimated signal power $\hat{P}(\boldsymbol{\mu})$.

To be able to describe more than one neuron, an SRF model class must contain parameters that can be adapted to each case. Ideally, the power of the model class to describe a population of neurons would be judged using parameters that produced models closest to the true SRFs (the *ideal models*), but we do not have *a priori* knowledge of those parameters. Instead, the parameters must be tuned in each case using the measured neural responses. One way to choose SRF model parameters is to minimize the mean squared error (MSE) between the neural response in the training data and the model prediction for the same stimulus; for example, the Wiener kernel minimizes the MSE for a model based on a finite impulse response filter of fixed length. This MSE is identical to the error power that would be obtained when the training data themselves are used as the reference measured response $\mathbf{r}$. Thus, by minimizing the MSE, we maximize the predictive power evaluated against the training data. The resulting maximum value, hereafter the *training predictive power*, will overestimate the predictive ability of the ideal model, since the minimum-MSE parameters will be overfit to the training data. (Overfitting is inevitable, because model estimates based on finite data will always capture some stimulus-independent response variability.) More precisely, the expected value of the training predictive power is an upper bound on the true predictive power of the model class; we therefore refer to the training predictive power itself as an *upper estimate* of the SRF model performance. We can also obtain a *lower estimate*, defined similarly, by empirically measuring the generalization performance of the model by cross-validation. This provides an unbiased estimate of the average generalization performance of the fitted models; however, since these models are inevitably overfit to their training data, the expected value of this *cross-validation predictive power* bounds the true predictive power of the ideal model from below, and thereby provides the desired lower estimate.

For any one recording, the predictive power of the ideal SRF model of a particular class can only be bracketed between these upper and lower estimates (that is, between the training and cross-validation predictive powers). As the noise in the recording grows, the model parameters will overfit more and more to the noise, and hence both estimates will grow looser. Indeed, in high-noise conditions, the model may primarily describe the stimulus-independent (noise) part of the training data, and so the training predictive power might exceed the estimated signal power ($\hat{P}(\boldsymbol{\mu})$), while the cross-validation predictive power may fall below zero (that is, the model's predictions may become more inaccurate than simply predicting a constant response). As such, the estimates may not usefully constrain the predictive power on a particular recording. However, assuming that the predictive power of a single model class is similar for a population of similar neurons, the noise dependence can be exploited to tighten the estimates when applied to the population as a whole, by extrapolating within the population to the zero noise point. This extrapolation allows us to answer the sort of question posed at the outset: how well, in an absolute sense, can a particular SRF model class account for the responses of a population of neurons?

## 4  Experimental Methods

Extracellular neural responses were collected from the primary auditory cortex of rodents during presentation of dynamic random chord stimuli. Animals (6 CBA/CaJ mice and 4 Long-Evans rats) were anaesthetized with either ketamine/medetomidine or sodium pentobarbital, and a skull fragment over auditory cortex was removed; all surgical and experimental procedures conformed to protocols approved by the UCSF Committee on Animal Research. An ear plug was placed in the left ear, and the sound field created by the free-field speakers was calibrated near the opening of the right pinna. Neural responses (205 recordings collected from 68 recording sites) were recorded in the thalamo-recipient layers

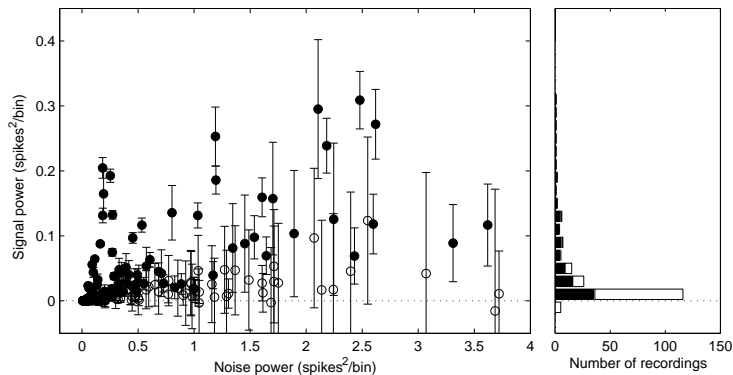

Figure 1: Signal power in neural responses.

of the left auditory cortex while the stimulus (see below) was presented to the right ear. Recordings often reflected the activity of a number of neurons; single neurons were identified by Bayesian spike-sorting techniques [13, 14] whenever possible. All analyses pool data from mice and rats, barbiturate and ketamine/medetomidine anesthesia, high and low frequency stimulation, and single-unit and multi-unit recordings; each group individually matched the aggregate behaviour described here.

The dynamic random chord stimulus used in the auditory experiments was similar to that used in a previous study [15], except that the intensity of component tone pulses was variable. Tone pulses were 20 ms in length, ramped up and down with 5 ms cosine gates. The times, frequencies and sound intensities of the pulses were chosen randomly and independently from 20 ms bins in time, 1/12 octave bins covering either 2–32 or 25–100 kHz in frequency, and 5 dB SPL bins covering 25–70 dB SPL in level. At any time point, the stimulus averaged two tone pulses per octave, with an expected loudness of approximately 73 dB SPL for the 2–32 kHz stimulus and 70 dB SPL for the 25–100 kHz stimulus. The total duration of each stimulus was 60 s. At each recording site, the 2–32 kHz stimulus was repeated 20 times, and the 25–100 kHz stimulus was repeated 10 times.

Neural responses were binned at 20 ms, and STRFs fit by linear regression of the average spike rate in each bin onto vectors formed from the amplitudes of tone pulses falling within the preceding 300 ms of the stimulus (15 pulse-widths, starting with pulses coincident with the target spike-rate bin). The regression parameters thus included a single filter weight for each frequency-time bin in this window, and an additional offset (or bias) weight. A Bayesian technique known as automatic relevance determination (ARD) [16] was used to improve the STRF estimates. In this case, an additional parameter reflecting the average noise in the response was also estimated. Models incorporating static output non-linearities were fit by kernel regression between the output of the linear model (fit by ARD) and the training data. The kernel employed was Gaussian with a half-width of 0.05 spike/bin; performance at this width was at least as good as that obtained by selecting widths individually for each recording by leave-one-out cross-validation. Cross-validation for lower estimates on model predictive power used 10 disjoint splits into 9/10 training data and 1/10 test data. Extrapolation of the predictive powers in the population, shown in Figs. 2 and 3, was performed using polynomial fits. The degree of the polynomial, determined by leave-one-out cross-validation, was quadratic for the lower estimates in Fig. 3 and linear in all other cases.

## 5    Results

We used the techniques described above to ask how accurate a description of auditory cortex responses could be provided by the STRF. Recordings were binned to match the

discretization rate of the stimulus and the signal power estimated using equation (1). Fig. 1 shows the distribution of signal powers obtained, as a scatter plot against the estimated noise power and as a histogram. The error bars indicate standard error intervals based on the estimated variances obtained from equation (2). A total of 92 recordings in the data set (42 from mouse, 50 from rat), shown by filled circles and histogram bars in Fig. 1, had signal power greater than one standard error above zero. The subsequent analysis was confined to these stimulus-responsive recordings.

For each such recording we estimated an STRF model by minimum-MSE linear regression, which is equivalent to obtaining the Wiener kernel for the time-series. The training predictive power of this model provided the upper estimate for the predictive power of the model class. The minimum-MSE solution generalizes poorly, and so generates overly pessimistic lower estimates in cross-validation. However, the linear regression literature provides alternative parameter estimation techniques with improved generalization ability. In particular, we used a Bayesian hyperparameter optimization technique known as Automatic Relevance Determination [16] (ARD) to find an optimized prior on the regression parameters, and then chose parameters which optimized the posterior distribution under this prior and the training data (this and other similar techniques are discussed in Sahani and Linden, "Evidence Optimization Techniques for Estimating Stimulus-Response Functions", this volume). The cross-validation predictive power of these estimates served as the lower estimates of the model class performance.

Fig. 2 shows the upper (○) and lower (●) estimates for the predictive power of the class of linear STRF models in our population of rodent auditory cortex recordings, as a function of the estimated noise level in each recording. The divergence of the estimates at higher noise levels, described above, is evident. At low noise levels the estimates do not converge perfectly, the extrapolated values being $0.4026 \pm 0.0076$ for the upper estimate and $0.1817 \pm 0.0079$ for the lower (intervals are standard errors). This gap is indicative of an SRF model class that is insufficiently powerful to capture the true stimulus-response relationship; even if noise were absent, the trained model from the class would only be able to approximate the true SRF in the region of the finite amount of data used for training, and so would perform better on those training data than on test data drawn from outside that region.

Fig. 3 shows the same estimates for simulations derived from linear fits to the cortical data. Simulated data were produced by generating Poisson spike trains with mean rates as predicted by the ARD-estimated models for real cortical recordings, and rectifying so that negative predictions were treated as zero. Simulated spike trains were then binned and analyzed in the same manner as real spike trains. Since the simulated data are spectrogram-linear by construction apart from the rectification, we expect the estimates to converge to a value very close to 1 with little separation. This result is evident in Fig. 3. Thus, the analysis correctly reports that virtually all of the response power in these simulations is linearly

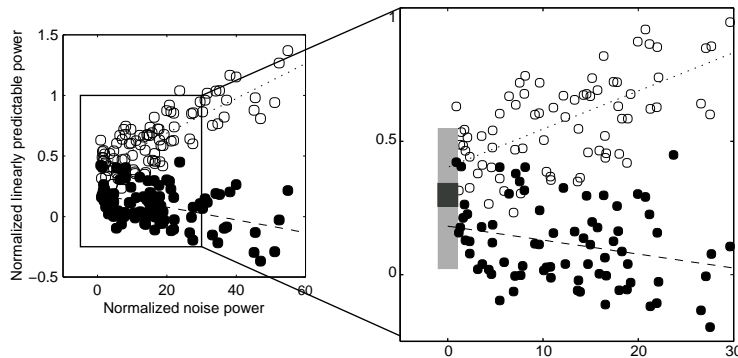

Figure 2: Evaluation of STRF predictive power in auditory cortex.

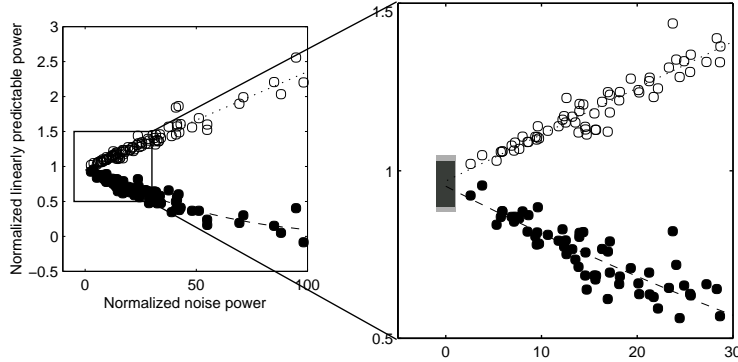

Figure 3: Evaluation of linearity in simulated data.

predictable from the stimulus spectrogram, attesting to the reliability of the extrapolated estimates for the real data in Fig. 2.

Some portion of the scatter of the points about the population average lines in Fig. 2 reflects genuine variability in the population, and so the extrapolated scatter at zero noise is also of interest. Intervals containing at least 50% of the population distribution for the cortical data are $(0.258, 0.548)$ for the upper estimate and $(0.023, 0.340)$ for the lower estimate (assuming normal scatter). These will be overestimates of the spread in the underlying population distribution because of additional scatter from estimation noise. The variability of STRF predictive power in the population appears unimodal, and the hypothesis that the distributions of the deviations from the regression lines are zero-mean normal in both cases cannot be rejected (Kolmogorov-Smirnov test, $\alpha = 0.1$). Thus the treatment of these recordings as coming from a single homogeneous population is reasonable. In Fig. 3, there is a small amount of downward bias and population scatter due to the varying amounts of rectification in the simulations; however, most of the observed scatter is due to estimation error resulting from the incorporation of Poisson noise.

The linear model is not constrained to predict non-negative firing rates. To test whether including a static output non-linearity could improve predictions, we also fit models in which the prediction from the ARD-derived STRF estimates was transformed time-point by time-point by a non-parametric non-linearity (see Experimental Methods) to obtain a new firing rate prediction. The resulting cross-validation predictive powers were compared to those of the spectrogram-linear model (data not shown). The addition of a static output non-linearity contributed very little to the predictive power of the STRF model class. Although the difference in model performance was significant ($p < 0.001$, Wilcoxon signed rank test), the mean normalized predictive power increase with the addition of a static output non-linearity was very small (0.031).

## 6 Conclusions

We have demonstrated a novel way to evaluate the fraction of response power in a population of neurons that can be captured by a particular class of SRF models. The confounding effects of noise on evaluation of model performance and estimation of model parameters are overcome by two key analytic steps. First, multiple measurements of neural responses to the same stimulus are used to obtain an unbiased estimate of the fraction of the response variance that is predictable in principle, against which the predictive power of a model may be judged. Second, Bayesian regression techniques are employed to lessen the effects of noise on linear model estimation, and the remaining noise-related bias is eliminated by exploiting the noise-dependence of parameter-estimation-induced errors in the predictive power to extrapolate model performance for a population of similar recordings to the zero

noise point. This technique might find broad applicability to regression problems in neuroscience and elsewhere, provided certain essential features of the data considered here are shared: repeated measurements must be made at the same input values in order to estimate the signal power; both inputs and repetitions must be numerous enough for the signal power estimate, which appears in the denominator of the normalized powers, to be well-conditioned; and finally we must have a group of different regression problems, with different normalized noise powers, that might be expected to instantiate the same underlying model class. Data with these features are commonly encountered in sensory neuroscience, where the sensory stimulus can be reliably repeated. The outputs modelled may be spike trains (as in the present study) or intracellular recordings; local-field, evoked-potential, or optical recordings; or even fMRI measurements.

Applying this technique to analysis of the primary auditory cortex we find that spectrogram-linear response components can account for only 18% to 40% (on average) of the power in extracellular responses to dynamic random chord stimuli. Further, elaborated models that append a static output non-linearity to the linear filter are barely more effective at predicting responses to novel stimuli than is the linear model class alone. Previous studies of auditory cortex have reached widely varying conclusions regarding the degree of linearity of neural responses. Such discrepancies may indicate that response properties are critically dependent on the statistics of the stimulus ensemble [6, 5, 10], or that cortical response linearity differs between species. Alternatively, as previous measures of linearity have been biased by noise, the divergent estimates might also have arisen from variation in the level of noise power across studies. Our approach represents the first evaluation of auditory cortex response predictability that is free of this potential noise confound. The high degree of response non-linearity we observe may well be a characteristic of all auditory cortical responses, given the many known non-linearities in the peripheral and central auditory systems [17]. Alternatively, it might be unique to auditory cortex responses to noisy sounds like dynamic random chord stimuli, or else may be general to all stimulus ensembles and all sensory cortices. Current and future work will need to be directed toward measurement of auditory cortical response linearity using different stimulus ensembles and in different species, and toward development of non-linear classes of models that predict auditory cortex responses more accurately than spectrogram-linear models.

## Footnotes

[1]An alternative would be to measure information or conditional entropy rates. However, the question of how much relevant information is preserved by a model is different from the question of how accurate a model's prediction is. For example, an information theoretic measure would not distinguish between a linear model and the same linear model cascaded with an invertible non-linearity.

# References

[1] Aertsen, A. M. H. J, Johannesma, P. I. M, & Hermes, D. J. (1980) *Biol. Cybern.* **38**, 235–248.

[2] Eggermont, J. J, Johannesma, P. M, & Aertsen, A. M. (1983) *Q Rev Biophys* **16**, 341–414.

[3] Kowalski, N, Depireux, D. A, & Shamma, S. A. (1996) *J. Neurophysiol.* **76**, 3524–3534.

[4] Shamma, S. A & Versnel, H. (1995) *Aud. Neurosci.* **1**, 255–270.

[5] Nelken, I, Rotman, Y, & Yosef, O. B. (1999) *Nature* **397**, 154–157.

[6] Rotman, Y, Bar-Yosef, O, & Nelken, I. (2001) *Hear. Res.* **152**, 110–127.

[7] Nelken, I, Prut, Y, Vaadia, E, & Abeles, M. (1994) *Hear. Res.* **72**, 206–222.

[8] Calhoun, B. M & Schreiner, C. E. (1998) *Eur J Neurosci* **10**, 926–940.

[9] Eggermont, J. J, Aertsen, A. M, & Johannesma, P. I. (1983) *Hear. Res.* **10**, 167–190.

[10] Theunissen, F. E, Sen, K, & Doupe, A. J. (2000) *J. Neurosci.* **20**, 2315–2331.

[11] Nelken, I, Prut, Y, Vaadia, E, & Abeles, M. (1994) *Hear. Res.* **72**, 223–236.

[12] Lindgren, B. W. (1993) *Statistical Theory.* (Chapman & Hall), 4th edition. ISBN: 0412041812.

[13] Lewicki, M. S. (1994) *Neural Comp* **6**, 1005–1030.

[14] Sahani, M. (1999) Ph.D. thesis (California Institute of Technology, Pasadena, California).

[15] deCharms, R. C, Blake, D. T, & Merzenich, M. M. (1998) *Science* **280**, 1439–1443.

[16] MacKay, D. J. C. (1994) *ASHRAE Transactions* **100**, 1053–1062.

[17] Popper, A & Fay, R, eds. (1992) *The Mammalian Auditory Pathway: Neurophysiology.* (Springer, New York).
